# High-Order Multi-Task Feature Learning to Identify Longitudinal Phenotypic Markers for Alzheimer's Disease Progression Prediction

**Hua Wang, Feiping Nie, Heng Huang,**
Department of Computer Science and Engineering,
University of Texas at Arlington, Arlington, TX 76019
{huawangcs, feipingnie}@gmail.com, heng@uta.edu

**Jingwen Yan, Sungeun Kim, Shannon L. Risacher, Andrew J. Saykin, Li Shen, for the ADNI**[*]
Department of Radiology and Imaging Sciences,
Indiana University School of Medicine, Indianapolis, IN 46202
{jingyan, sk31, srisache, asaykin, shenli}@iupui.edu

## Abstract

Alzheimer's disease (AD) is a neurodegenerative disorder characterized by progressive impairment of memory and other cognitive functions. Regression analysis has been studied to relate neuroimaging measures to cognitive status. However, whether these measures have further predictive power to infer a trajectory of cognitive performance over time is still an under-explored but important topic in AD research. We propose a novel high-order multi-task learning model to address this issue. The proposed model explores the temporal correlations existing in imaging and cognitive data by structured sparsity-inducing norms. The sparsity of the model enables the selection of a small number of imaging measures while maintaining high prediction accuracy. The empirical studies, using the longitudinal imaging and cognitive data of the ADNI cohort, have yielded promising results.

## 1 Introduction

Neuroimaging is a powerful tool for characterizing neurodegenerative process in the progression of Alzheimer's disease (AD). Neuroimaging measures have been widely studied to predict disease status and/or cognitive performance [1, 2, 3, 4, 5, 6, 7]. However, whether these measures have further predictive power to infer a trajectory of cognitive performance over time is still an under-explored yet important topic in AD research. A simple strategy typically used in longitudinal studies (*e.g.*, [8]) is to analyze a single summarized value such as average change, rate of change, or slope. This approach may be inadequate to distinguish the complete dynamics of cognitive trajectories and thus become unable to identify underlying neurodegenerative mechanism. Figure 1 shows a schematic example. Let us look at the plot of Cognitive Score 2. The red and blue groups can be easily separated by their complete trajectories. However, given very similar score values at the time points of t0 and t3, any of the aforementioned summarized values may not be sufficient to identify the group difference. Therefore, if longitudinal cognitive outcomes are available, it would be beneficial to use the complete information for the identification of relevant imaging markers [9, 10].

[*]Data used in preparation of this article were obtained from the Alzheimer's Disease Neuroimaging Initiative (ADNI) database (adni.loni.ucla.edu). As such, the investigators within the ADNI contributed to the design and implementation of ADNI and/or provided data but did not participate in analysis or writing of this report. A complete listing of ADNI investigators can be found at: http://adni.loni.ucla.edu/wp-content/uploads/how_to_apply/ADNI_Acknowledgement_List.pdf.

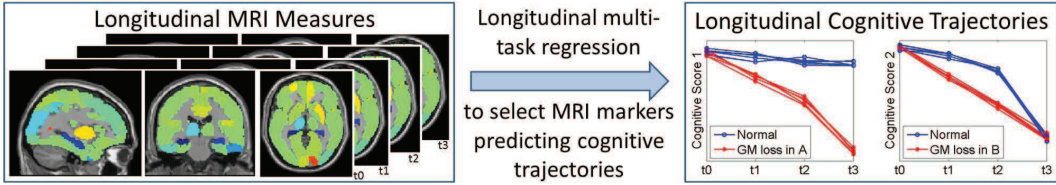

Figure 1: Longitudinal multi-task regression of cognitive trajectories on MRI measures.

However, how to identify the temporal imaging features that predict longitudinal outcomes is a challenging machine learning problem. First, the input data and response measures often are high-order tensors, not regular data/label matrix. For example, both input neuroimaging measures (samples × features × time) and output cognitive scores (samples × scores × time) are 3D tensors. Thus, it is not trivial to build the longitudinal learning model for tensor data. Second, the associations between features and a specific task (*e.g.* cognitive score) at two consecutive time points are often correlated. How to efficiently include such correlations of associations cross time is unclear. Third, some longitudinal learning tasks are often interrelated to each other. For example, it is well known that [3, 4] in RAVLT assessment, the total number of words remembered by the participants in the first 5 learning trials heavily impacts the total number of words which can be recalled in the 6th learning trial, and the results of these two measures both partially determines the final recognition rate after 30 minutes delay. How to integrate such tasks correlations into longitudinal learning model is under-explored.

In this paper, we focus on the problem of predicting longitudinal cognitive trajectories using neuroimaging measures. We propose a novel high-order multi-task feature learning approach to identify longitudinal neuroimaging markers that can accurately predict cognitive scores over all the time points. The sparsity-inducing norms are introduced to integrate the correlations existing in both features and tasks. As a result, the selected imaging markers can fully differentiate the entire longitudinal trajectory of relevant scores and better capture the associations between imaging markers and cognitive changes over time. Because the structured sparsity-inducing norms enforce the correlations along two directions of the learned coefficient tensor, the parameters in different sparsity norms are tangled together by distinct structures and lead to a difficult optimization problem. We derive an efficient algorithm to solve the proposed high-order multi-task feature learning objective with closed form solution in each iteration. We further prove the global convergence of our algorithm. We apply the proposed longitudinal multi-task regression method to the ADNI cohort. In our experiments, the proposed method not only achieves competitive prediction accuracy but also identifies a small number of imaging markers that are consistent with prior knowledge.

## 2 High-Order Multi-Task Feature Learning Using Sparsity-Inducing Norms

For AD progression prediction using longitudinal phenotypic markers, the input imaging features are a set of matrices $\mathcal{X} = \{X_1, X_2, \ldots, X_T\} \in \mathbb{R}^{d \times n \times T}$ corresponding to the measurements at $T$ consecutive time points, where $X_t$ is the phenotypic measurements for a certain type of imaging markers, such as voxel-based morphometry (VBM) markers (see details in Section 3) used in this study, at time $t \, (1 \le t \le T)$. Obviously, $\mathcal{X}$ is a tensor data with $d$ imaging features, $n$ subject samples and $T$ time points. The output cognitive assessments for the same set of subjects are a set of matrices $\mathcal{Y} = \{Y_1, Y_2, \ldots, Y_T\} \in \mathbb{R}^{n \times c \times T}$ for a certain type of the cognitive measurements, such as RAVLT memory scores (see details in Section 3), at the same $T$ consecutive time points. Again, $\mathcal{Y}$ is a tensor data with $n$ samples, $c$ scores, and $T$ time points. Our goal is to learn from $\{\mathcal{X}, \mathcal{Y}\}$ a model that can reveal the longitudinal associations between the imaging and cognitive trajectories, by which we expect to better understand how the variations of different regions of human brains affect the AD progression, such that we can improve the diagnosis and treatment to the disease.

Prior regression analyses typically study the associations between imaging features and cognitive measures at each time point separately, which is equivalent to assume that the learning tasks, *i.e.*, cognitive measures, at different time points are independent. Although this assumption can simplify the problem and make the solution easier to obtain, it overlooks the temporal correlations of imaging and cognitive measures. To address this, we propose to jointly learn a single longitudinal regression model for the all time points to identify imaging markers which are associated to cog-

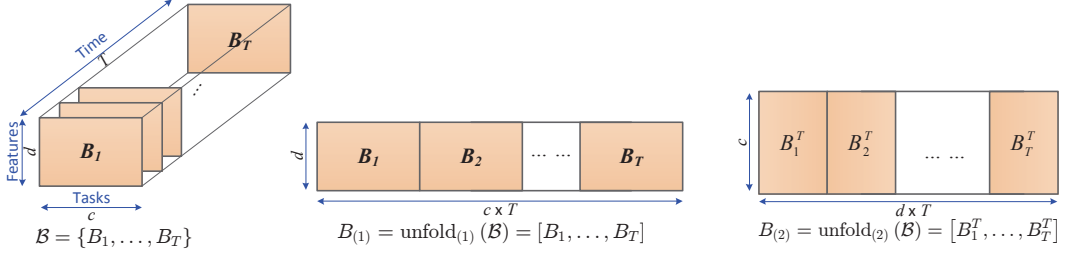

Figure 2: **Left**: visualization of the coefficient tensor $\mathcal{B}$ learned for the association study on longitudinal data. **Middle**: the matrix unfolded from $\mathcal{B}$ along the first mode (feature dimension). **Right**: the matrix unfolded from $\mathcal{B}$ along the second mode (task dimension).

nitive patterns. As a result, we aim to learn a coefficient tensor (a stack of coefficient matrices) $\mathcal{B} = \{B_1, \cdots, B_n\} \in \mathbb{R}^{d \times c \times T}$, as illustrated in the left panel of Figure 2, to reveal the temporal changes of the coefficient matrices. Given the additional time dimension, our problem becomes a difficult high-order data analysis problem, which we call as *high-order multi-task learning*.

## 2.1 Longitudinal Multi-Task Feature Learning

In order to associate the imaging markers and the cognitive measures, the multivariate regression model was used in traditional association studies, which minimizes the following objective:

$$\min_{\mathcal{B}} \ J_0 = \left\| \mathcal{B} \otimes_1 \mathcal{X}^T - \mathcal{Y} \right\|_F^2 + \alpha \left\| \mathcal{B} \right\|_2^2 = \sum_{t=1}^{T} ||X_t^T B_t - Y_t||_F^2 + \alpha \sum_{t=1}^{T} \sum_{k=1}^{d} ||\mathbf{b}_t^k||_2^2 \ . \tag{1}$$

where $\mathbf{b}_t^k$ denotes the $k$-th row of coefficient matrix $B_t$ at time $t$. Apparently, the objective $J_0$ in Eq. (1) can be decoupled for each individual time point. Therefore it does not take into account the longitudinal correlations between imaging features and cognitive measures. Because our goal in the association study is to select the imaging markers which are connected to the temporal changes of all the cognitive measures, the $T$ groups of regression tasks at different time points should not be decoupled and have to be performed simultaneously. To achieve this, we select imaging markers correlated to all the cognitive measures at all time points by introducing the sparse regularization [11, 12, 13] into the longitudinal data regression and feature selection model as follows:

$$\min_{\mathcal{B}} \ J_1 = \sum_{t=1}^{T} ||X_t^T B_t - Y_t||_F^2 + \alpha \sum_{k=1}^{d} \sqrt{\sum_{t=1}^{T} ||\mathbf{b}_t^k||_2^2} = \sum_{t=1}^{T} ||X_t^T B_t - Y_t||_F^2 + \alpha \left\| B_{(1)} \right\|_{2,1} \ , \tag{2}$$

where we denote $\mathrm{unfold}_k (\mathcal{B}) = B_{(k)} \in \mathbb{R}^{I_k \times (I_1 \ldots I_{k-1} I_{k+1} \ldots I_n)}$ as the unfolding operation to a general $n$-mode tensor $\mathcal{B}$ along the $k$-th mode, and $B_{(1)} = \mathrm{unfold}_1 (\mathcal{B}) = [B_1, \ldots, B_T]$ as illustrated in the middle panel of Figure 2. By solving the objective $J_1$, the imaging features with common influences across all the time points for all the cognitive measures will be selected due to the second term in Eq. (2), which is a tensor extension of the widely used $\ell_{2,1}$-norm for matrix.

## 2.2 High-Order Multi-Task Correlations

The objective $J_1$ in Eq. (2) couples all the learning tasks together, which, though, still does not address the correlations among different learning tasks at different time points. As discussed earlier, during the AD progression, many cognitive measures are interrelated together and their effects during the process could overlap, thus it is necessary to further develop the objective $J_1$ in Eq. (2) to leverage the useful information conveyed by the correlations among different cognitive measures. In order to capture the longitudinal patterns of the AD data, we consider two types of tasks correlations. First, for an individual cognitive measure, although its association to the imaging features at different stages of the disease could be different, its associations patterns at two consecutive time points tend to be similar [9]. Second, we know that [4, 14] during the AD progression, different cognitive measures are interrelated to each other. Mathematically speaking, the above two types of correlations can both be described by the low ranks of the coefficient matrices unfolded from the

coefficient tensor along different modes. Thus we further develop our learning model in Eq. (2) to impose additional low rank regularizations to exploit these task correlations.

Let $B_{(2)} = \mathrm{unfold}_2 (\mathcal{B}) = \left[ B_1^T, \ldots, B_T^T \right]$ as illustrated in the right panel of Figure 2, we minimize the ranks of $B_{(1)}$ and $B_{(2)}$ to capture the two types of task correlations, one for each type, as follows:

$$\min_{\mathcal{B}} \; J_2 = \sum_{t=1}^{T} ||X_t^T B_t - Y_t||_F^2 + \alpha \left\| B_{(1)} \right\|_{2,1} + \beta \left( \left\| B_{(1)} \right\|_* + \left\| B_{(2)} \right\|_* \right) \; , \tag{3}$$

where $\|\cdot\|_*$ denote the trace norm of a matrix. Given a matrix $M \in \mathbb{R}^{n \times m}$ and its singular values $\sigma_i \, (1 \leq i \leq \min(n, m))$, the trace norm of $M$ is defined as $\|M\|_* = \sum_{i=1}^{\min(n,m)} \sigma_i = \mathrm{Tr} \left( M M^T \right)^{\frac{1}{2}}$. It has been shown that [15, 16, 17] the trace-norm is the best convex approximation of the rank-norm. Therefore, the third and fourth terms of $J_2$ in Eq. (3) indeed minimize the rank of the unfolded learning model $\mathcal{B}$, such that the two types of correlations among the learning tasks at different time points can be utilized. Due to its capabilities for both imaging marker selection and task correlation integration on longitudinal data, we call $J_2$ defined in Eq. (3) as the proposed *High-Order Multi-Task Feature Learning* model, by which we will study the problem of longitudinal data analysis to predict cognitive trajectories and identify relevant imaging markers.

## 2.3 New Optimization Algorithm and Its Global Convergence

Despite its nice properties, our new objective $J_2$ in Eq. (3) is a non-smooth convex problem. Some existing methods can solve it, but not efficiently. Thus, in this subsection we will derive a new efficient algorithm to solve this optimization problem with global convergence proof, where we employ an iteratively reweighted method [18] to deal with the non-smooth regularization terms.

Taking the derivative of the objective $J_2$ in Eq. (3) with respect to $B_t$ and set it as 0, we obtain[1]:

$$2 X_t X_t^T B_t - 2 X_t Y_t + 2 \alpha D B_t + 2 \beta \left( \bar{D} B_t + B_t \hat{D} \right) = 0 \; , \tag{4}$$

where $D$ is a diagonal matrix with $D\,(i, i) = \frac{1}{2 \sqrt{\sum_{t=1}^{T} \left\| \mathbf{b}_t^k \right\|_2^2}}$, $\bar{D} = \frac{1}{2} \left( B_{(1)} B_{(1)}^T \right)^{-1/2}$ and $\hat{D} = \frac{1}{2} \left( B_{(2)} B_{(2)}^T \right)^{-1/2}$. We can re-write Eq. (4) as following:

$$\left( X_t X_t^T + \alpha D + \beta \bar{D} \right) B_t + \beta B_t \hat{D} = X_t Y_t \; , \tag{5}$$

which is a Sylvester equation and can be solved in closed form. When the time $t$ changes from 1 to $T$, we can calculate $B_t \, (1 \leq t \leq T)$ by solving Eq. (5). Because $D$, $\bar{D}$ and $\hat{D}$ are dependent on $\mathcal{B}$ and can be seen as latent variables, we propose an iterative algorithm to obtain the global optimum solutions of $B_t \, (1 \leq t \leq T)$, which is summarized in Algorithm 1.

**Convergence analysis of the new algorithm.** We first prove the following two useful lemmas, by which we will prove the convergence of Algorithm 1.

**Lemma 1** *Given a constant $\alpha > 0$, for function $f(x) = x - \frac{x^2}{2\alpha}$, we have $f(x) \leq f(\alpha)$ for any $x \in \mathbb{R}$. The equality holds if and only if $x = \alpha$.*

The proof of Lemma 1 is obvious and skipped due to space limit.

**Lemma 2** *Given two semi-positive definite matrices $A$ and $\tilde{A}$, the following inequality holds:*

$$\textbf{tr} \left( \tilde{A}^{\frac{1}{2}} \right) - \frac{1}{2} \textbf{tr} \left( \tilde{A} A^{-\frac{1}{2}} \right) \leq \textbf{tr} \left( A^{\frac{1}{2}} \right) - \frac{1}{2} \textbf{tr} \left( A A^{-\frac{1}{2}} \right) \; . \tag{6}$$

*The equality holds if and only if $A = \tilde{A}$.*

**Algorithm 1:** A new algorithm to solve the optimization problem in Eq. (3).

---

**Data**: $\mathcal{X} = [X_1, X_2, \ldots, X_T] \in \mathbb{R}^{d \times n \times T}, \mathcal{Y} = [Y_1, Y_2, \ldots, Y_T] \in \mathbb{R}^{n \times c \times T}$.

**1**. Set $g = 1$. Initialize $B_t^{(1)} \in \mathbb{R}^{d \times c}$ ($1 \leq t \leq T$) using the linear regression results at each individual time point.

**repeat**

> **2**. Calculate the diagonal matrix $D^{(g)}$, where the $i$-th diagonal element is computed as $D^{(g)}(i, i) = \frac{1}{2\sqrt{\Sigma_{t=1}^{T}\left\|\mathbf{b}_t^{(g),k}\right\|_2^2}}$;
>
> calculate $\bar{D}^{(g)} = \frac{1}{2}\left(B_{(1)}^{(g)}\left(B_{(1)}^{(g)}\right)^T\right)^{-\frac{1}{2}}$; calculate $\hat{D}^{(g)} = \frac{1}{2}\left(B_{(2)}^{(g)}\left(B_{(2)}^{(g)}\right)^T\right)^{-\frac{1}{2}}$.
>
> **3**. Update $B_t^{(g+1)}$ ($1 \leq t \leq T$) by solving the Sylvester equation in Eq. (5).
> **4**. $g = g + 1$.

**until** *Converges*

**Result**: $\mathcal{B} = [B_1, B_2, \ldots, B_T] \in \mathbb{R}^{d \times c \times T}$.

---

**Proof**: Because $A$ and $\tilde{A}$ are two semi-positive definite matrices and we know that $\mathbf{tr}\left(A\tilde{A}\right) = \mathbf{tr}\left(\tilde{A}A\right)$, we can derive:

$$
\begin{aligned}
\mathbf{tr}\left(A^{\frac{1}{2}} - 2\tilde{A}^{\frac{1}{2}} + \tilde{A}A^{-\frac{1}{2}}\right) = \mathbf{tr}\left(A^{-\frac{1}{4}}\left(A + \tilde{A} - A^{\frac{1}{2}}\tilde{A}^{\frac{1}{2}} - \tilde{A}^{\frac{1}{2}}A^{\frac{1}{2}}\right)A^{-\frac{1}{4}}\right) = \\
\mathbf{tr}\left(A^{-\frac{1}{4}}\left(A^{\frac{1}{2}} - \tilde{A}^{\frac{1}{2}}\right)^2 A^{-\frac{1}{4}}\right) = \left\|A^{-\frac{1}{4}}\left(A^{\frac{1}{2}} - \tilde{A}^{\frac{1}{2}}\right)\right\|_F^2 \geq 0 ,
\end{aligned}
\tag{7}
$$

by which we have the following inequality $\mathbf{tr}\left(\tilde{A}^{\frac{1}{2}}\right) - \frac{1}{2}\mathbf{tr}\left(\tilde{A}A^{-\frac{1}{2}}\right) \leq \frac{1}{2}\mathbf{tr}\left(A^{\frac{1}{2}}\right)$, which is equivalent to Eq. (6) and completes the proof of Lemma 2. $\square$

Now we prove the convergence of Algorithm 1, which is summarized by the following theorem.

**Theorem 1** *Algorithm 1 monotonically decreases the objective of the problem in Eq.* (3) *in each iteration, and converges to the globally optimal solution.*

**Proof**: In Algorithm 1, we denote the updated $B_t$ in each iteration as $\tilde{B}_t$. We also denote the least square loss in the $g$-th iteration as $\mathcal{L}^{(g)} = \sum_{t=1}^{T} ||X_t^T B_t^{(g)} - Y_t||_F^2$. According to Step 3 of Algorithm 1 we know that the following inequality holds:

$$
\begin{aligned}
\mathcal{L}^{(g+1)} + \alpha \sum_{t=1}^{T} \mathbf{tr}\left(\tilde{B}_t^T D\tilde{B}_t\right) + \beta \sum_{t=1}^{T} \mathbf{tr}\left(\tilde{B}_t^T \bar{D}\tilde{B}_t\right) + \beta \sum_{t=1}^{T} \mathbf{tr}\left(\tilde{B}_t \hat{D}\tilde{B}_t^T\right) \leq \\
\mathcal{L}^{(g)} + \alpha \sum_{t=1}^{T} \mathbf{tr}\left(B_t^T DB_t\right) + \beta \sum_{t=1}^{T} \mathbf{tr}\left(B_t^T \bar{D}B_t\right) + \beta \sum_{t=1}^{T} \mathbf{tr}\left(B_t \hat{D}B_t^T\right) .
\end{aligned}
\tag{8}
$$

Denote the updated $B_{(1)}$ as $\tilde{B}_{(1)}$, and the updated $B_{(2)}$ as $\tilde{B}_{(1)}$, from Eq. (8) we can derive:

$$
\begin{aligned}
\mathcal{L}^{(g+1)} + \alpha\,\mathbf{tr}\left(\tilde{B}_{(1)}^T D\tilde{B}_{(1)}\right) + \beta\,\mathbf{tr}\left(\tilde{B}_{(1)}\tilde{B}_{(1)}^T \bar{D}\right) + \beta\,\mathbf{tr}\left(\tilde{B}_{(2)}\tilde{B}_{(2)}^T \hat{D}\right) \leq \\
\mathcal{L}^{(g)} + \alpha \sum_{t=1}^{T} \mathbf{tr}\left(B_{(1)}^T DB_{(1)}\right) + \beta \sum_{t=1}^{T} \mathbf{tr}\left(B_{(1)}B_{(1)}^T \bar{D}\right) + \beta \sum_{t=1}^{T} \mathbf{tr}\left(B_{(2)}B_{(2)}^T \hat{D}\right) .
\end{aligned}
\tag{9}
$$

According to the definitions of $D$, $\bar{D}$ and $\hat{D}$, we have:

$$
\begin{aligned}
\mathcal{L}^{(g+1)} + \frac{\alpha}{2} \sum_{k=1}^{d} \frac{\sum_{t=1}^{T}||\mathbf{b}_t^{(g+1),k}||_2^2}{\sqrt{\sum_{t=1}^{T}||\mathbf{b}_t^{(g),k}||_2^2}} + \frac{\beta}{2}\,\mathbf{tr}\left(\tilde{B}_{(1)}\tilde{B}_{(1)}^T\left(B_{(1)}B_{(1)}^T\right)^{-\frac{1}{2}}\right) + \frac{\beta}{2}\,\mathbf{tr}\left(\tilde{B}_{(2)}\tilde{B}_{(2)}^T\left(B_{(2)}B_{(2)}^T\right)^{-\frac{1}{2}}\right) \leq \\
\mathcal{L}^{(g)} + \frac{\alpha}{2} \sum_{k=1}^{d} \frac{\sum_{t=1}^{T}||\mathbf{b}_t^{(g),k}||_2^2}{\sqrt{\sum_{t=1}^{T}||\mathbf{b}_t^{(g),k}||_2^2}} + \frac{\beta}{2}\,\mathbf{tr}\left(B_{(1)}B_{(1)}^T\left(B_{(1)}B_{(1)}^T\right)^{-\frac{1}{2}}\right) + \frac{\beta}{2}\,\mathbf{tr}\left(B_{(1)}B_{(1)}^T\left(B_{(2)}B_{(2)}^T\right)^{-\frac{1}{2}}\right) .
\end{aligned}
\tag{10}
$$

Then according to Lemma 1 and Lemma 2, the following three inequalities hold:

$$
\sqrt{\sum_{t=1}^{T}||\mathbf{b}_t^{(g+1),k}||_2^2} - \frac{\sum_{t=1}^{T}||\mathbf{b}_t^{(g+1),k}||_2^2}{2\sqrt{\sum_{t=1}^{T}||\mathbf{b}_t^{(g),k}||_2^2}} \leq \sqrt{\sum_{t=1}^{T}||\mathbf{b}_t^{(g),k}||_2^2} - \frac{\sum_{t=1}^{T}||\mathbf{b}_t^{(g),k}||_2^2}{2\sqrt{\sum_{t=1}^{T}||\mathbf{b}_t^{(g),k}||_2^2}} .
\tag{11}
$$

$$\mathbf{tr}\left(\tilde{B}_{(1)}\tilde{B}_{(1)}^T\right) - \mathbf{tr}\left(\frac{1}{2}\tilde{B}_{(1)}\tilde{B}_{(1)}^T\left(B_{(1)}B_{(1)}^T\right)^{-\frac{1}{2}}\right) \le \mathbf{tr}\left(B_{(1)}B_{(1)}^T\right) - \mathbf{tr}\left(\frac{1}{2}B_{(1)}B_{(1)}^T\left(B_{(1)}B_{(1)}^T\right)^{-\frac{1}{2}}\right),$$
(12)

$$\mathbf{tr}\left(\tilde{B}_{(2)}\tilde{B}_{(2)}^T\right) - \mathbf{tr}\left(\frac{1}{2}\tilde{B}_{(2)}\tilde{B}_{(2)}^T\left(B_{(2)}B_{(2)}^T\right)^{-\frac{1}{2}}\right) \le \mathbf{tr}\left(B_{(2)}B_{(2)}^T\right) - \mathbf{tr}\left(\frac{1}{2}B_{(2)}B_{(2)}^T\left(B_{(2)}B_{(2)}^T\right)^{-\frac{1}{2}}\right).$$
(13)

Adding the both sides of of Eqs. (10–13) together, we can obtain:

$$\mathcal{L}^{(g+1)} + \alpha \sum_{k=1}^{d} \sqrt{\sum_{t=1}^{T} ||\mathbf{b}_t^{(g+1),k}||_2^2} + \beta\,\mathbf{tr}\left(\tilde{B}_{(1)}\tilde{B}_{(1)}^T\right) + \beta\,\mathbf{tr}\left(\tilde{B}_{(2)}\tilde{B}_{(2)}^T\right) \le$$

$$\mathcal{L}^{(g+1)} + \alpha \sum_{k=1}^{d} \sqrt{\sum_{t=1}^{T} ||\mathbf{b}_t^{(g),k}||_2^2} + \beta\,\mathbf{tr}\left(B_{(1)}B_{(1)}^T\right) + \beta\,\mathbf{tr}\left(B_{(2)}B_{(2)}^T\right)$$
(14)

Thus, our algorithm decreases the objective value of Eq. (3) in each iteration. When the objective value keeps unchange, Eq. (4) is satisfied, *i.e.*, the K.K.T. condition of the objective is satisfied. Thus, our algorithm reaches one of the optimal solutions. Because the objective in Eq. (3) is a convex problem, Algorithm 1 will converge to one of the globally optimal solution. □

## 3  Experiments

We evaluate the proposed method by applying it to the Alzheimer's Disease Neuroimaging Initiative (ADNI) cohort to examine the association between a wide range of imaging measures and two types of cognitive measures over a certain period of time. Our goal is to discover a compact set of imaging markers that are closely related to cognitive trajectories.

**Imaging markers and cognitive measures.** Data used in this work were obtained from the ADNI database (`adni.loni.ucla.edu`). One goal of ADNI has been to test whether serial MRI, PET, other biological markers, and clinical and neuropsychological assessment can be combined to measure the progression of Mild Cognitive Impairment (MCI) and early AD. For up-to-date information, see `www.adni-info.org`. We downloaded 1.5 T MRI scans and demographic information for 821 ADNI-1 participants. We performed voxel-based morphometry (VBM) on the MRI data by following [8], and extracted mean modulated gray matter (GM) measures for 90 target regions of interest (ROIs) (see Figure 3 for the ROI list and detailed definitions of these ROIs in [3]). These measures were adjusted for the baseline intracranial volume (ICV) using the regression weights derived from the healthy control (HC) participants at the baseline. We also downloaded the longitudinal scores of the participants in two independent cognitive assessments including Fluency Test and Rey's Auditory Verbal Learning Test (RAVLT). The details of these cognitive assessments can be found in the ADNI procedure manuals[2]. The time points examined in this study for both imaging markers and cognitive assessments included baseline (BL), Month 6 (M6), Month 12 (M12) and Month 24 (M24). All the participants with no missing BL/M6/M12/M24 MRI measurements and cognitive measures were included in this study. A total of 417 subjects were involved in our study, including 84 AD, and 191 MCI and 142 HC participants. We examined 3 RAVLT scores RAVLT_TOTAL, RAVLT_TOT6 and RAVLT_RECOG, and 2 Fluency scores FLU_ANIM and FLU_VEG.

### 3.1  Improved Cognitive Score Prediction from Longitudinal Imaging Markers

We first evaluate the proposed method by applying it to the ADNI cohort for predicting the two types of cognitive scores using the VBM markers, tracked over four different time points. Our goal in this experiment is to improve the prediction performance.

**Experimental setting.** We compare the proposed method against its two close counterparts including multivariate linear regression (LR) and ridge regression (RR). LR is the simplest and widely used regression model in statistical learning and brain image analysis. RR is a regularized version of LR to avoid over-fitting. Due to their mathematical nature, these two methods are performed for

Table 1: Performance comparison for memory score prediction measured by RMSE.

|  | LR | RR | TGL | Ours ($\ell_{2,1}$-norm only) | Ours (trace norm only) | Ours |
|---|---|---|---|---|---|---|
| RAVLT | 0.380 | 0.341 | 0.318 | 0.306 | 0.301 | **0.283** |
| Fluency | 0.171 | 0.165 | 0.155 | 0.144 | 0.147 | **0.135** |

each cognitive measure at each time point separately, and thus they cannot make use of the temporal correlation. We also compare our method to a recent longitudinal method, called as Temporal Group Lasso Multi-Task Regression (TGL) [9]. TGL takes into account the longitudinal property of the data, which, however, is designed to analyze only one single memory score at a time. In contrast, besides imposing structured sparsity via tensor $\ell_{2,1}$-norm regularization for imaging marker selection, our new method also imposes two trace norm regularizations to capture the interrelationships among different cognitive measures over the temporal dimension. Thus, the proposed method is able to perform association study for all the relevant scores of a cognitive test at the same time, *e.g.*, our method can simultaneously deal with the three RAVLT scores, or the two Fluency scores.

To evaluate the usefulness of each component of the proposed method, we implement three versions of our method as follows. First, we only impose the $\ell_{2,1}$-norm regularization on the unfolded coefficient tensor $\mathcal{B}$ along the feature mode, denoted as "$\ell_{2,1}$-norm only". Second, we only impose the trace norm regularizations on the two coefficient matrices unfolded from the coefficient tensor $\mathcal{B}$ along the feature and task modes respectively, denoted as "trace norm only". Finally, we implement the full version of our new method that solves the proposed objective in Eq. (3). Note that, if no regularization is imposed, our method is degenerated to the traditional LR method.

To measure prediction performance, we use standard 5-fold cross-validation strategy by computing the root mean square error (RMSE) between the predicted and actual values of the cognitive scores on the testing data only. Specifically, the whole set of subjects are equally and randomly partitioned into five subsets, and each time the subjects within one subset are selected as the testing samples and all other subjects in the remaining four subsets are used for training the regression models. This process is repeated for five times and average results are reported in Table 1. To treat all regression tasks equally, data for each response variable is normalized to have zero mean and unit variance.

**Experimental results.** From Table 1 we can see that the proposed method is consistently better than the three competing methods, which can be attributed to the following reasons. First, because LR and RR methods by nature can only deal with one individual cognitive measure at one single time point at a time, they cannot benefit from the correlations across different cognitive measures over the entire time course. Second, although TGL method improves the previous two methods in that it does take into account longitudinal data patterns, it still assumes all the test scores (*i.e.*, learning tasks) from one cognitive assessment to be independent, which, though, is not true in reality. For example, it is well known that [3, 4] in RAVLT assessment, the total number of words remembered by the participants in the first 5 learning trials (RAVLT_TOTAL) heavily impacts the total number of words which can be recalled in the 6th learning trial (RAVLT_TOT6), and the results of these two measures both partially determines the final recognition rate after 30 minutes delay (RAVLT_RECOG). In contrast, our new method considers all $c$ learning tasks ($c = 3$ for RAVLT assessment and $c = 2$ for Fluency assessment) as an integral learning object as formulated in Eq. (3), such that their correlations can be incorporated by the two imposed low-rank regularization terms.

Besides, we also observe that the two degenerated versions of the proposed method do not perform as well as their full version counterpart, which provides a concrete evidence to support the necessities of the component terms of our learning objective in Eq. (3) and justifies our motivation to impose $\ell_{2,1}$-norm regularization for feature selection and trace norm regularization to capture task correlations.

## 3.2 Identification of Longitudinal Imaging Markers

Because one of the primary goals of our regression analysis is to identify a subset of imaging markers which are highly correlated to the AD progression reflected by the cognitive changes over time. Therefore, we examine the imaging markers identified by the proposed methods with respect to the longitudinal changes encoded by the cognitive scores recorded at the four consecutive time points.

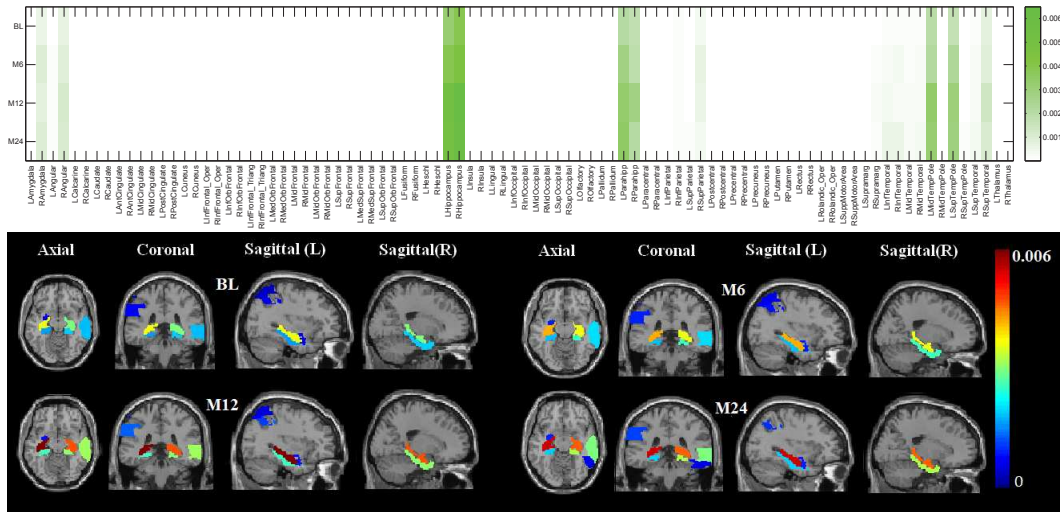

Figure 3: Top panel: Average regression weights of imaging markers for predicting three RAVLT memory scores. Bottom panel: Top 10 average weights mapped onto the brain.

Shown in Figure 3 are (1) the heat map of the learned weights (magnitudes of the average regression weights for all three RAVLT scores at each time point) of the VBM measures at different time points calculated by our method; and (2) the top 10 weights mapped onto the brain anatomy. A first glance at the heat map in Figure 3 indicates that the selected imaging markers have clear patterns that span across all the four studied time points, which demonstrates that these markers are longitudinally stable and thereby can potentially serve as screening targets over the course of AD progression.

Moreover, we observe that the bilateral hippocampi and parahippocampal gyri are among the top selected features. These findings are in accordance with the known knowledge that in the pathological pathway of AD, medial temporal lobe is firstly affected, followed by progressive neocortical damage [19, 20]. Evidence of a significant atrophy of middle temporal region in AD patients has also been observed in previous studies [21, 22, 23].

In summary, the identified longitudinally stable imaging markers are highly suggestive and strongly agree with the existing research findings, which warrants the correctness of the discovered imaging-cognition associations to reveal the complex relationships between MRI measures and cognitive scores. This is important for both theoretical research and clinical practices for a better understanding of AD mechanism.

## 4 Conclusion

To reveal the relationship between longitudinal cognitive measures and neuroimaging markers, we have proposed a novel high-order multi-task feature learning model, which selects the longitudinal imaging markers that can accurately predict cognitive measures at all the time points. As a result, these imaging markers could fully differentiate the entire longitudinal trajectory of relevant cognitive measures and better capture the associations between imaging markers and cognitive changes over time. To solve our new objective, which uses the non-smooth structured sparsity-inducing norms, we have derived an iterative algorithm with a closed form solution in each iteration. We have further proved our algorithm converges to the global optimal solution. The validations using ADNI imaging and cognitive data have demonstrated the promise of our method.

**Acknowledgement.** This work was supported by NSF CCF-0830780, CCF-0917274, DMS-0915228, and IIS-1117965 at UTA; and by NSF IIS-1117335, NIH R01 LM011360, UL1 RR025761, U01 AG024904, RC2 AG036535, R01 AG19771, and P30 AG10133-18S1 at IU. Data used in the work were obtained from the ADNI database. ADNI funding information is available at http://adni.loni.ucla.edu/wp-content/uploads/how_to_apply/ADNI_DSP_Policy.pdf.

## Footnotes

[1] $\|M\|_{2,1}$ is a non-smooth function of $M$ and not differentiable when one of its row $\mathbf{m}^i = 0$. Following [18], we introduce a small perturbation $\zeta > 0$ to replace $\|M\|_{2,1}$ by $\sum_i \sqrt{\left\| \mathbf{m}^i \right\|_2^2 + \zeta}$, which is smooth and differentiable with respect to $M$. Apparently, $\sum_i \sqrt{\left\| \mathbf{m}^i \right\|_2^2 + \zeta}$ is reduced to $\|M\|_{2,1}$ when $\zeta \to 0$. In the sequel of this paper, we implicitly apply this replacement for all $\|\cdot\|_{2,1}$. Following the same idea, we also introduce a small perturbation $\xi > 0$ to replace $\|M\|_*$ by $\textbf{tr} \left( M M^T + \xi I \right)^{\frac{1}{2}}$ for the same reason.

[2]`http://www.adni-info.org/Scientists/ProceduresManuals.aspx`

# References

[1] C Hinrichs, V Singh, G Xu, SC Johnson, and ADNI. Predictive markers for ad in a multi-modality framework: an analysis of mci progression in the adni population. *Neuroimage*, 55(2):574–89, 2011.

[2] CM Stonnington, C Chu, S Kloppel, and et al. Predicting clinical scores from magnetic resonance scans in alzheimer's disease. *Neuroimage*, 51(4):1405–13, 2010.

[3] L. Shen, S. Kim, and *et al*. Whole genome association study of brain-wide imaging phenotypes for identifying quantitative trait loci in MCI and AD: A study of the ADNI cohort. *Neuroimage*, 2010.

[4] H. Wang, F. Nie, H. Huang, S. Risacher, C. Ding, A.J. Saykin, L. Shen, et al. Sparse multi-task regression and feature selection to identify brain imaging predictors for memory performance. In *ICCV*, 2011.

[5] D. Zhang and D. Shen. Multi-modal multi-task learning for joint prediction of multiple regression and classification variables in alzheimer's disease. *Neuroimage*, 2011.

[6] H. Wang, F. Nie, H. Huang, S. Kim, Nho K., S. Risacher, A. Saykin, and L. Shen. Identifying Quantitative Trait Loci via Group-Sparse Multi-Task Regression and Feature Selection: An Imaging Genetics Study of the ADNI Cohort. *Bioinformatics*, 28(2):229–237, 2012.

[7] H. Wang, F. Nie, H. Huang, S. Risacher, A. Saykin, and L. Shen. Identifying Disease Sensitive and Quantitative Trait Relevant Biomarkers from Multi-Dimensional Heterogeneous Imaging Genetics Data via Sparse Multi-Modal Multi-Task Learning. *Bioinformatics*, 28(18):i127–i136, 2012.

[8] S. L. Risacher, L. Shen, J. D. West, S. Kim, B. C. McDonald, L. A. Beckett, D. J. Harvey, Jr. Jack, C. R., M. W. Weiner, A. J. Saykin, and ADNI. Longitudinal MRI atrophy biomarkers: relationship to conversion in the ADNI cohort. *Neurobiol Aging*, 31(8):1401–18, 2010.

[9] J. Zhou, L. Yuan, J. Liu, and J. Ye. A multi-task learning formulation for predicting disease progression. In *SIGKDD*, 2011.

[10] H. Wang, F. Nie, H. Huang, J. Yan, S. Kim, Nho K., S. Risacher, A. Saykin, and L. Shen. From Phenotype to Genotype: An Association Study of Candidate Phenotypic Markers to Alzheimer's Disease Relevant SNPs. *Bioinformatics*, 28(12):i619–i625, 2012.

[11] A. Argyriou, T. Evgeniou, and M. Pontil. Multi-task feature learning. *NIPS*, pages 41–48, 2007.

[12] G. Obozinski, B. Taskar, and M. Jordan. Multi-task feature selection. *Technical report, Department of Statistics, University of California, Berkeley*, 2006.

[13] M. Yuan and Y. Lin. Model selection and estimation in regression with grouped variables. *Journal of The Royal Statistical Society Series B*, 68(1):49C–67, 2006.

[14] H. Wang, F. Nie, H. Huang, S. Risacher, A. Saykin, and L. Shen. Identifying ad-sensitive and cognition-relevant imaging biomarkers via joint classification and regression. *Medical Image Computing and Computer-Assisted Intervention (MICCAI 2011)*, pages 115–123, 2011.

[15] B. Recht, M. Fazel, and P.A. Parrilo. Guaranteed minimum-rank solutions of linear matrix equations via nuclear norm minimization. *Arxiv preprint arxiv:0706.4138*, 2007.

[16] E.J. Candès and B. Recht. Exact matrix completion via convex optimization. *Foundations of Computational Mathematics*, 9(6):717–772, 2009.

[17] E.J. Candes and T. Tao. The power of convex relaxation: Near-optimal matrix completion. *Information Theory, IEEE Transactions on*, 56(5):2053–2080, 2010.

[18] I.F. Gorodnitsky and B.D. Rao. Sparse signal reconstruction from limited data using focuss: A reweighted minimum norm algorithm. *Signal Processing, IEEE Transactions on*, 45(3):600–616, 1997.

[19] H. Braak and E. Braak. Neuropathological stageing of alzheimer-related changes. *Acta neuropathologica*, 82(4):239–259, 1991.

[20] A. Delacourte, JP David, N. Sergeant, L. Buee, A. Wattez, P. Vermersch, F. Ghozali, C. Fallet-Bianco, F. Pasquier, F. Lebert, et al. The biochemical pathway of neurofibrillary degeneration in aging and alzheimers disease. *Neurology*, 52(6):1158–1158, 1999.

[21] L.G. Apostolova, P.H. Lu, S. Rogers, R.A. Dutton, K.M. Hayashi, A.W. Toga, J.L. Cummings, and P.M. Thompson. 3d mapping of mini-mental state examination performance in clinical and preclinical alzheimer disease. *Alzheimer Disease & Associated Disorders*, 20(4):224, 2006.

[22] A. Convit, J. De Asis, MJ De Leon, CY Tarshish, S. De Santi, and H. Rusinek. Atrophy of the medial occipitotemporal, inferior, and middle temporal gyri in non-demented elderly predict decline to Alzheimer's disease. *Neurobiol of aging*, 21(1):19–26, 2000.

[23] V. Julkunen, E. Niskanen, S. Muehlboeck, M. Pihlajamäki, M. Könönen, M. Hallikainen, M. Kivipelto, S. Tervo, R. Vanninen, A. Evans, et al. Cortical thickness analysis to detect progressive mild cognitive impairment: a reference to alzheimer's disease. *Dementia and geriatric cognitive disorders*, 28(5):404–412, 2009.

